# A Hidden Markov Model for de Novo Peptide Sequencing

**Bernd Fischer, Volker Roth, Joachim M. Buhmann**
Institute of Computational Science
ETH Zurich
CH-8092 Zurich, Switzerland
`bernd.fischer@inf.ethz.ch`

**Jonas Grossmann, Sacha Baginsky,**
**Wilhelm Gruissem**
Institute of Plant Sciences
ETH Zurich
CH-8092 Zurich, Switzerland

**Franz Roos,**
**Peter Widmayer**
Inst. of Theoretical Computer Science
ETH Zurich
CH-8092 Zurich, Switzerland

## Abstract

*De novo* Sequencing of peptides is a challenging task in proteome research. While there exist reliable DNA-sequencing methods, the high-throughput *de novo* sequencing of proteins by mass spectrometry is still an open problem. Current approaches suffer from a lack in precision to detect mass peaks in the spectrograms. In this paper we present a novel method for de novo peptide sequencing based on a hidden Markov model. Experiments effectively demonstrate that this new method significantly outperforms standard approaches in matching quality.

## 1 Introduction

The goal of *de novo* peptide sequencing is to reconstruct an amino acid sequence from a given mass spectrum. *De novo* sequencing by means of mass spectrometry is a very challenging task, since many practical problems like measurement errors or peak suppression have to be overcome. It is, thus, not surprising that current approaches to reconstruct the sequence from mass spectra are usually limited to those species for which genome information is available. This case is a simplified problem of the *de novo* sequencing problem, since the hypothesis space of possible sequences is restricted to the known ones contained in a sequence database.

In this paper we present a Hidden Markov Model (HMM) for *de novo* sequencing. The main difference to standard methods which are all based on dynamic programming [2, 1] lies in the fully probabilistic model. Our trained HMM defines a *generative model* for mass spectra which, for instance, is used for scoring *observed* spectra according to their likelihood given a peptide sequence. Besides predicting the most likely sequence, however, the HMM framework is far more general in the sense that it additionally allows us to specify the *confidence* in the predictions.

# 2 Tandem Mass Spectrometry

In a typical sequencing experiment by mass spectrometry a protein is digested with the help of an enzyme. This digestion reaction breaks the protein into several *peptides*, each of which consists of a short sequence of typically 10 to 20 amino acid residues, with an additional H-atom at the N-terminus and an OH-group at the C-terminus.

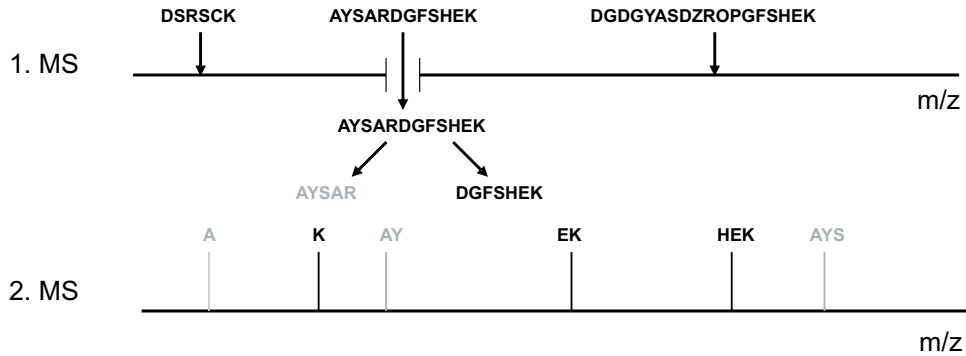

Figure 1: In the first mass measurement the parent mass is selected. In the second measurement the peptide is dissociated and the mass of the ion fragments is measured.

There are two measurement steps in a *tandem mass spectrometer*. The first step is responsible for filtering peptides of a certain total mass (also called the *parent mass*). The difficulty in measuring the parent mass arises from different $^{12}C/^{13}C$ isotope proportions of the approximately 30-80 C-atoms contained in a peptide. Fluctuations of the $^{13}C$ fraction result in a binomial distribution of parent masses in the measurement. Given such an "ion count distribution" one can roughly estimate the mono-isotopic parent mass of the peptide, where the term mono-isotopic here refers to a peptide that contains exclusively $^{12}C$ atoms. In practice, all isotope configurations of a peptide with parent masses that do not exceed the estimated mono-isotopic mass by more than a predefined offset are separated from the other peptides and passed to the second spectrometer.

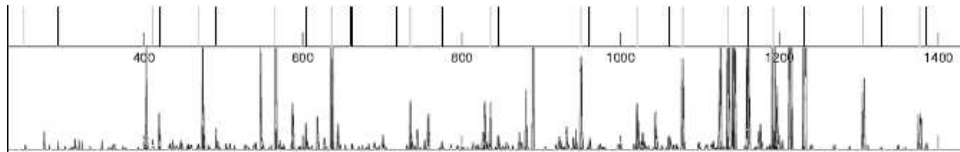

Figure 2: Top: The ideal peaks of a peptide sequence are drawn. Bottom: The spectrum of the corresponding peptide.

In the second mass measurement, a peptide is split into two fragments by means of collision induced dissociation with a noble gas. In almost all cases the peptide is broken between two amino acids. Thus, an ideal spectrum is composed of the masses of all prefix and suffix sequences of the peptide. Deviations from this ideal case are e.g. caused by problems in determining the exact mono-isotopic mass of the fragments due to isotope shifts. Further complications are caused by an accidental loss of water ($H_2O$), ammonia ($NH_3$) or other molecules in the collision step. Moreover, the ion counts are not uniformly distributed over the spectrum. And last but not least, the measurements are noisy.

## 3 The Hidden Markov Model for de Novo Peptide Sequencing

A peptide can formally be described as a sequence of symbols from a fixed alphabet $\mathcal{A}$ of 20 amino acids. We will denote amino acids with $\alpha \in \mathcal{A}$ and the mass of an amino acid with $M(\alpha)$. The input data is a spectrum of ion counts over all mass units. The ion count for mass $m$ is denoted by $x(m)$. The spectra are discretized to approximately one Dalton mass units and normalized such that the mean ion count per Dalton is constant.

The mono-isotopic parent mass $m'_p$ of the peptide $\mathcal{P} = (\alpha_1, \ldots, \alpha_n)$ with $\alpha_i \in \mathcal{A}$ is the sum of all amino acid masses plus a constant mass for the N- and C-termini. $m'_p = constN + \sum_{i=1}^{n} M(\alpha_i) + constC$. For the sake of simplicity it is assumed that the N- and C-termini are not present and thus the parent mass considered in the sequel is

$$m_p = \sum_{i=1}^{n} M(\alpha_i) \,. \tag{1}$$

In the HMM framework a spectrum is regarded as a realization of a random process. The *physical* process that generates spectra is based on the fact that a peptide is randomly broken into two parts by interaction with a noble gas. Each of these parts is detected in the mass spectrometer and increases the ion-count in the corresponding mass interval. Finally, a histogram over many such events is measured. In order to derive a *model* of the generation process, we make the simplifying assumptions that (i) breaks occur only at amino acid boundaries, and (ii) the probability of observing a break after a certain amino acid depends only on the amino acid itself. These assumptions allow us to model the generative process by way of a *Markov process* on a finite state automation. In such a model, the process of generating a spectrum for a peptide of known parent mass is formalized as a path through the automaton in 1 Dalton steps until the constraint on the parent mass is satisfied.

### 3.1 Finite State Automaton

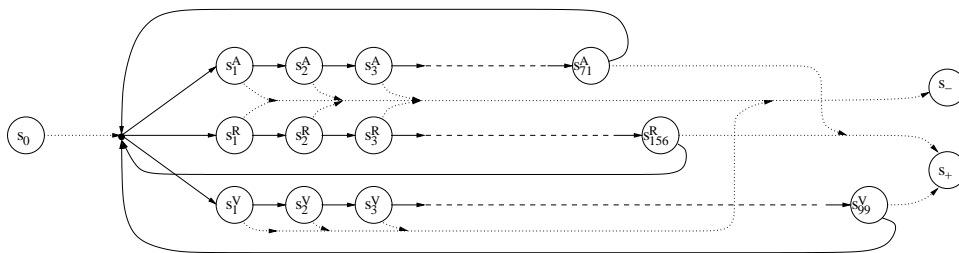

Figure 3: The finite state machine of the Hidden Markov Model. For each amino acid $\alpha$ there is a list of $M(\alpha)$ states.

The finite state automaton (fig. 3) has one initial state $s_0$. For each amino acid $\alpha \in \mathcal{A}$ there exists a list of $M(\alpha)$ states $s_1^\alpha, \ldots, s_{M(\alpha)}^\alpha$. Together with the end states $s_+$ and $s_-$ the complete set of states is

$$\mathcal{S} = \{s_0\} \cup \left\{ s_j^\alpha \mid \alpha \in \mathcal{A}, 1 \leq j \leq M(\alpha) \right\} \cup \{s_+, s_-\} \,. \tag{2}$$

The bold edges in the graph correspond to state transition probabilities $a(s, t)$ from state $s$ to state $t$. Once the automation is in the first state $s_1^\alpha$ of a state list of one amino acid $\alpha$, it has to pass through all other states within the specific list. Thus for the next $M(\alpha)$ steps the list corresponding to amino acid $\alpha$ is linearly traversed. If the automaton is in the last state $s_{M(\alpha)}^\alpha$ of a list, it can reach the start states $s_1^{\alpha'}$ of any other amino acid $\alpha'$. The random variable for the state sequence is denoted by $Y_1, \ldots, Y_{m_p}$. The transition probabilities are

$$a(s, t) = P\{Y_{i+1} = t | Y_i = s\} = \begin{cases} 1 & \forall \alpha \in \mathcal{A}, 1 \leq i < M(\alpha): s = s_i^\alpha \wedge t = s_{i+1}^\alpha \\ r_\alpha & \forall \alpha \in \mathcal{A}, \beta \in \mathcal{A}: s = s_{m(\beta)}^\beta \wedge t = s_1^\alpha \\ 0 & else \,. \end{cases} \tag{3}$$

The first row ($a(s,t) = 1$) describes the case where the automaton is in a non-terminating state of a list of amino acid $\alpha$ ($1 \leq i < M(\alpha) : s = s_i^\alpha$), where the following state is accepted with probability 1. The second row, on the contrary, refers to a terminating state of a list. In such a case, the starting state of any other amino acid is selected with probability $r_\alpha$. The probabilities $r_\alpha$ are the probabilities of occurrence of amino acid $\alpha$.

The transition probabilities $a(s_0, t)$ from the start state $s_0$ are the occurrence probabilities of the amino acids.

$$a(s_0, t) = \begin{cases} r_\alpha & \forall \alpha \in \mathcal{A} : t = s_1^\alpha \\ 0 & else \end{cases} \tag{4}$$

Finally one has to ensure that the parent mass constraint is fulfilled. In order to satisfy the constraint we device a time dependent hidden Markov model in which the transition probability changes with a heavy side function at time $m_p$ from $a(s,t)$ to $a'(s,t)$. The dotted arrows in figure 3 show the transition probabilities $a'(s,t)$ into the end states $s_+$ and $s_-$.

$$a'(s,t) = \begin{cases} 1 & \forall \alpha \in \mathcal{A} : s = s_{M(\alpha)}^\alpha, t = s_+ \\ 1 & \forall \alpha \in \mathcal{A}, 1 \leq i < M(\alpha) : s = s_i^\alpha, t = s_- \\ 0 & else \end{cases} \tag{5}$$

If the automaton is in the last state $s_{M(\alpha)}^\alpha$ of an amino acid state list, it changes to the positive end state $s_+$ with probability 1 since the parent mass constraint is satisfied. If the automaton is in one of the other states, it changes to the negative end state $s_-$ since the parent mass constraint is violated. It is important to realize that all amino acid sequences that fulfill the parent mass constraint can be transformed into state sequences that end in the positive state $s_+$ and vice versa.

## 3.2 Emission Probabilities

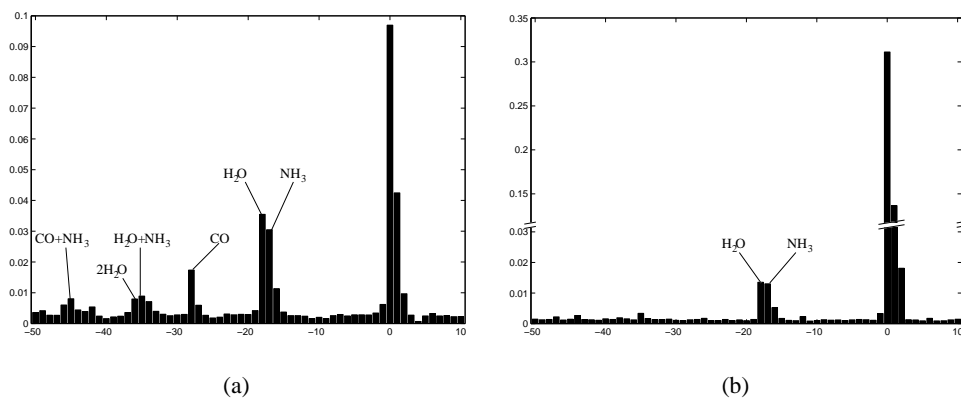

Figure 4: Mean height of ion counts for different shifts with respect to the ideal prefix fragments (a) and suffix fragments (b).

At each state of the finite state automaton an ion count value is emitted. Figure 4 shows the mean ion count for different positions relative to the amino acid bound averaged over all amino acids. The histograms are taken over the training examples described in the experimental section. It happens quite frequently that an amino acid looses water ($H_2O$) or ammonia ($NH_3$). The ion count patterns for the prefix fragments (fig. 4 a) and the suffix fragments (fig. 4 b) are quite different due to chemical reasons. For instance, carbon

monoxide loss in the suffix fragments is an unlikely event. Suffix fragments are more stable than prefix fragments: the central peak at position 0 (amino acid boundary) is three times higher for the suffix fragments than for the prefix fragments. Note that in figure 4 b) we used two different scales.

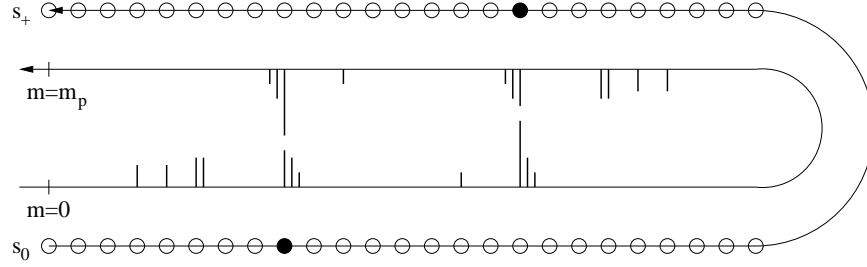

Figure 5: Folding the spectrum in the middle makes the intern mirror symmetry of the problem visible. The Markov chain models a sequence with three amino acids. The filled circles correspond to the amino acid boundaries. Each amino acid bound generates an ion count pattern for the prefix fragment and one for the suffix fragment.

Breaking a peptide in the second mass spectrometer produces both a prefix and a suffix fragment. To simultaneously process peaks of both types of fragments, we use one forward and one backward Markov chain which are independent of each other. Due to the inherent mirror symmetry of the problem (fig. 5) it is sufficient to limit the length of both models to $m_p/2$. For the recognition process we assume that we simultaneously observe two peaks $\mathbf{x}_{m,1} = x(m)$ and $\mathbf{x}_{m,2} = x(m_p - m)$ in step $m$. The joint observation of the prefix and the suffix peaks is an essential modeling step in our method.

The forward and the backward Markov chains are extended to hidden Markov models to describe the ion counts in the mass spectra. The emission probabilities depend on the two states of the prefix and suffix sequence, since these states give rise to ion counts in the measurements. We define

$$b_{s,s'}(\mathbf{x}_m) = P\left\{ \bar{X}_m = \mathbf{x}_m = (x(m), x(m_p - m)) \mid \bar{Y}_m = (s, s') \right\} \qquad (6)$$

as the emission probabilities of ion counts.

$\bar{X}_m$ are the (coupled) random variables of the ion counts. The hidden variables for the state sequence are denoted by $\bar{Y}_m$. This notion of coupled variables $\bar{X}_m$ describes the transition from two independent Markov chains to one coupled hidden Markov model with a squared number of states (2-tuple states).

The joint probability of observable and hidden variables given the parent mass $m_p$ is

$$P\left\{ X = \mathbf{x}, Y = y \mid s_+, m_p \right\} = a\left(s_0, y_1\right) a'\left(y_{m_p}, s_+\right) \cdot \qquad (7)$$
$$\cdot \left[ \prod_{m=1}^{\frac{m_p-1}{2}} b_{y_m, y_{m_p-m}}(\mathbf{x}_m) a\left(y_m, y_{m+1}\right) a\left(y_{m_p-m}, y_{m_p-m+1}\right) \right] a\left( y_{\frac{m_p-1}{2}}, y_{\frac{m_p-1}{2}+1} \right)$$

This formula holds for parent masses with an odd Dalton value, an equivalent formula can be derived for the even case. The first term in eq. (7) is the joint probability from $s_0$ to $y_1$ in the prefix model and the transition $y_{m_p}$ to $s_+$ in the suffix model. In each term of the product, two peaks are observed on both sides of the spectrum: one at position $m$ and the other at the mirror position $m_p - m$. The joint probability of emissions is defined by $b_{y_m, y_{m_p-m}}(x_m, x_{m_p-m})$. Furthermore, the transition probabilities of the prefix and suffix sequences are multiplied which reflects the independence assumption of the Markov model.

The two chains are connected by the transition probability $a(y_{(m_p-1)/2}, y_{(m_p-1)/2+1})$ of traversing from the last state of the forward Markov chain to the first state of the backward chain.

### 3.3 Most Probable Sequence

The input spectrum usually comes with an estimate of the parent mass with a tolerance of about 1 Dalton. Using a maximum likelihood approach the parent mass estimate is

$$\hat{m}_p = \operatorname*{argmax}_{m_p} P\{X = \mathbf{x} \mid s_+, m_p\} = \operatorname*{argmax}_{m_p} \sum_y P\{X = \mathbf{x}, Y = y \mid s_+, m_p\} . \quad (8)$$

The sum over all sequences can be computed efficiently by dynamic programming using the forward algorithm.

One result of *de novo* peptide sequencing is the computation of the best sequence generating a given spectrum. Given the estimated parent mass $\hat{m}_p$ the maximum posterior estimate of the sequence is

$$y^* = \operatorname*{argmax}_{y} P\{Y = y \mid X = \mathbf{x}, s_+, \hat{m}_p\} = \operatorname*{argmax}_{y} P\{X = \mathbf{x}, Y = y \mid s_+, \hat{m}_p\} .$$
$$(9)$$

The best sequence can efficiently be found by the Viterbi algorithm. To compute the posterior probability one has to normalize the joint probability $P\{X = \mathbf{x}, Y = y \mid s_+, \hat{m}_p\}$ by the evidence $P\{X = \mathbf{x} \mid s_+, \hat{m}_p\}$ using the forward-backward algorithm.

In the mass spectra ions with very low mass or almost parent mass are less frequently observed than ions with a medium mass. Therefore it becomes quite difficult to estimate the whole sequence with a high score. It is also possible to give a score for each subsequence of the peptide, especially a score for each amino acid. An amino acid is a subsequence $y_p, \ldots, y_q$ of the state sequence $y_1, \ldots, y_{m_p}$.

$$P\{y_p, \ldots, y_q \mid s_+, x, m_p\} \quad (10)$$

$$= \frac{\sum_{y_1, \ldots y_{p-1}} \sum_{y_{q+1}, \ldots, y_{m_p}} P\{y_1, \ldots, y_{m_p}, x \mid s_+, m_p\}}{P\{x \mid s_+, m_p\}} \quad (11)$$

This can again be computed by some variation of the forward and backward algorithm.

### 3.4 Simplification of the Model

The coupled hidden Markov model has $2\,375^2 = 5\,640\,625$ states that leads to a runtime of 20 minutes per peptide which for practical applications is problematic. A significant simplification is achieved by assuming that there are two spectra observed, where the second one is the mirror version of the first one. The emission probabilities in this simplified model only depend on the states of the prefix Markov chain (fig. 6). Thus the emission of mirror peaks $x(m_p - m)$ is deterministically coupled to the emission of the peak $x_m$. Since this model has only $2\,375$ states, the computation time reduces to 1-2 seconds per peptide.

## 4 Experiments

In our experiments a protein probe of plant cell vacuoles (*Arabidopsis thaliana*) was digested with trypsin. The mass spectrometer gave an output of 7056 different candidate spectra. From a database search with SEQUEST [3] and further validation with PeptideProphet [4], 522 spectra with a confidence larger than $90\%$ were extracted. It was shown that the PeptideProphet score is a very reliable scoring method for peptide identification by database search. The database output was used as training data. The

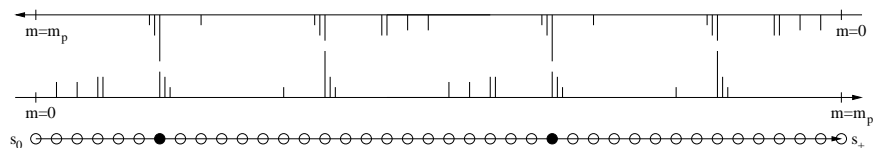

Figure 6: In the simplified model two mirrored spectra are observed. The emission of symbols is coupled with the amino acid bounds of the prefix sequence.

quality of the HMM inference is measured by the ratio of common amino acid boundaries and the number of amino acids in the database sequence. The performance of the HMM was tested by leave-one-out cross validation: in each training step the emission probabilities and the amino acid occurrence probabilities are re-estimated, with one sequence excluded from the training set. To estimate the emission probabilities, the ion count is discretized to a fixed number of bins, in such a way that all bins contain an equal number of counts. The leave-one-out scheme is repeated for different numbers of discretization levels.

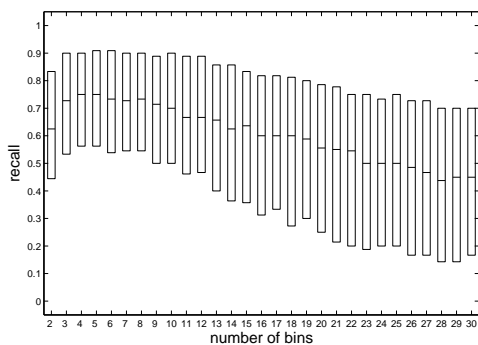

Figure 7: Cross validation of recall rates for different number of bins in the discretization process. Depicted are the lower quartile, the median and the upper quartile.

The resulting performance recall rate are depicted in figure 7. Choosing 5 bins yields the highest recall value.

We have chosen the prominent *de novo* sequencing programs LUTEFISK [6] and PEAKS [5] as competitors for the simplified HMM. We compared the sequence from the HMM with the highest scoring sequences from the other programs. In figure 8 a) the estimated parent masses compared to the database parent mass is drawn. The plot demonstrates that all *de novo* sequencing methods tend to overestimate the parent mass. The best one is the HMM with 89.1% correct estimations, whereas only 59.3% of the LUTEFISK estimates and 58.1% of the PEAKS estimates are correct. In figure 8 b) boxplots of the recognition rate of peak positions is drawn. The three lines in the box correspond to the lower quartile, the median and the upper quartile of the distribution. The median recall of the HMM is 75.0%, for Lutefisk 53.9% and for Peaks 56.7%. Note that the lower quartile of the HMM results is above 50%, whereas it is below 10% for the other programs.

## 5 Conclusion and Further Work

A novel method for the analysis of mass spectra in *de novo* peptide sequencing is presented in this paper. The proposed hidden Markov model is a fully probabilistic model for the generation process of mass spectra. The model was tested on mass spectra from vacuola

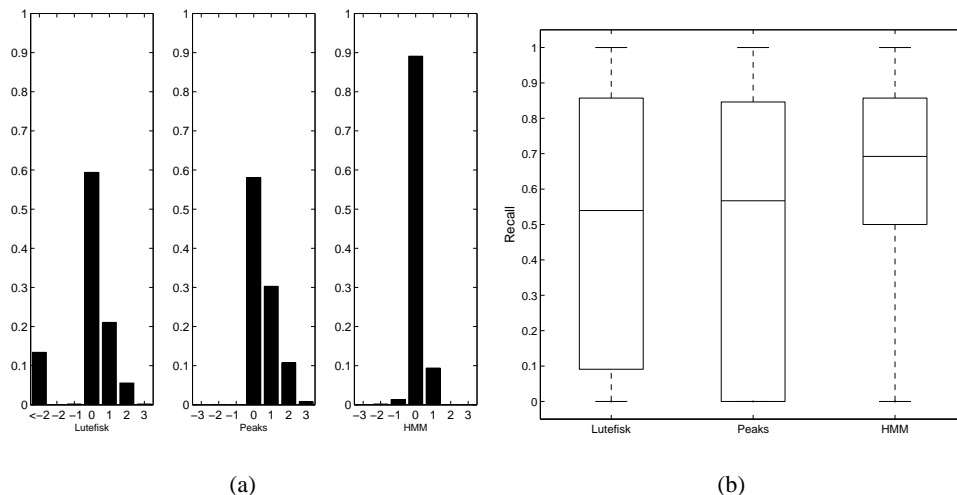

|     | (a) |     | (b) |
| --- | --- | --- | --- |

Figure 8: a) Histogram on difference of estimated parent mass and database output. b) Recall of peak positions.

proteins. The HMM clearly outperforms its competitors in recognition of the parent mass and peak localization. In further work additional model parameters will be introduced to represent and to detect amino acids with post-translational modifications. Reliable subsequences can further be used for a tagged database search to identify peptides with post-translational modifications. Our method shows a large potential for high throughput *de novo* sequencing of proteins which is unmatched by competing techniques.

**Acknowledgment** This work has been partially supported by DFG grant # Buh 914/5.

# References

[1] Sacha Baginsky, Mark Cieliebak, Wilhelm Gruissem, Torsten Kleffmann, Zsuzsanna Lipták, Matthias Müller, and Paolo Penna. Audens: A tool for automatic de novo peptide sequencing. Technical Report 383, ETH Zurich, Dept. of Computer Science, 2002.

[2] Ting Chen, Ming-Yang Kao, Matthew Tepel, John Rush, and George M. Church. A dynamic programming approach to de novo peptide sequencing via tandem mass spectrometry. *Journal of Computational Biology*, 8(3):325–337, 2001.

[3] Jimmy K. Eng, Ashley L. McCormack, and John R. Yates. An approach to correlate tandem mass spectral data of peptides with amino acid sequences in a protein database. *American Society for Mass Spectrometry*, 5(11):976–989, 1994.

[4] Andrew Keller, Alexey I. Nesvizhskii, Eugene Kolker, and Ruedi Aebersold. Empirical statistical model to estimate the accuracy of peptide identifications made by MS/MS and database search. *Analytical Chemistry*, 2002.

[5] Bin Ma, Kaizhong Zhang, Christopher Hendrie, Chengzhi Liang, Ming Li, Amanda Doherty-Kirby, and Gilles Lajoie. Peaks: Powerful software for peptide de novo sequencing by tandem mass spectrometry. *Rapid Communication in Mass Spectrometry*, 17(20):2337–2342, 2003.

[6] J. Alex Taylor and Richard S. Johnson. Implementation and uses of automated de novo peptide sequencing by tandem mass spectrometry. *Analytical Chemistry*, 73:2594–2604, 2001.
